# Boosting Structured Prediction
# for Imitation Learning

**Nathan Ratliff, David Bradley, J. Andrew Bagnell, Joel Chestnutt**
Robotics Institute
Carnegie Mellon University
Pittsburgh, PA 15213
{ndr, dbradley, dbagnell, joel.chestnutt}@ri.cmu.edu

## Abstract

The Maximum Margin Planning (MMP) (Ratliff et al., 2006) algorithm solves imitation learning problems by learning linear mappings from features to cost functions in a planning domain. The learned policy is the result of minimum-cost planning using these cost functions. These mappings are chosen so that example policies (or trajectories) given by a teacher appear to be lower cost (with a loss-scaled margin) than any other policy for a given planning domain. We provide a novel approach, MMPBOOST , based on the functional gradient descent view of boosting (Mason et al., 1999; Friedman, 1999a) that extends MMP by "boosting" in new features. This approach uses simple binary classification or regression to improve performance of MMP imitation learning, and naturally extends to the class of structured maximum margin prediction problems. (Taskar et al., 2005) Our technique is applied to navigation and planning problems for outdoor mobile robots and robotic legged locomotion.

## 1 Introduction

"Imitation learning" of control or navigational behaviors is important in many application areas. Recently, (Ratliff et al., 2006) demonstrated that imitation learning of long horizon and goal-directed behavior can be naturally formulated as a structured prediction problem over a space of policies or system trajectories. In this work, the authors demonstrate that efficient planning algorithms (e.g. for deterministic systems or general Markov Decision Problems) can be taught to generalize a set of examples provided by a supervisor. In essence, the algorithm attempts to linearly combine features into costs so that the resulting cost functions make demonstrated example policies appear optimal by a margin over all other policies. The technique utilizes the idea that while a desired behavior or control strategy is often quite clear to a human expert, hand designing cost functions that induce this behavior may be difficult.

Unfortunately, this Maximum Margin Planning (MMP) approach, as well as related techniques for maximum margin structured learning developed in (Taskar et al., 2005) and (Taskar et al., 2003), depend on linearly combining a prespecified set of features. [1] Adopting a new variant of the general ANYBOOST algorithm described in (Mason et al., 1999) or similarly (Friedman, 1999a), we propose an alternate extension to Maximum Margin Planning specifically, and maximum margin structured learning generally, in which we perform subgradient descent in the space of cost functions rather than within any fixed parameterization. In this way, we show that we can "boost in" new features using simple classification that help us a solve a more difficult structured prediction problem. The

application of boosting to structured learning techniques was first explored in (Dietterich et al., 2004), within the context of boosting Conditional Random Fields. This paper extends that result to maximum-margin techniques, and provides a more general functional gradient derivation.

We then demonstrate three applications of our technique. First, using only smoothed versions of an overhead image as input, we show that MMPBOOST is able to match human navigation performance well on the task of navigating outdoor terrain. Using the same input, linear MMP, by contrast, performs almost no better than straight line paths. Next we demonstrate that we can develop a local obstacle detection/avoidance control system for an autonomous outdoor robot by observing an expert teleoperator drive. Finally, we demonstrate on legged locomotion problems the use a slow but highly accurate planner to train a fast, approximate planner using MMPBOOST .

## 2   Preliminaries

We model, as in (Ratliff et al., 2006), planning problems as discrete Markov Decision Processes. Let $s$ and $a$ index the state and action spaces $\mathcal{S}$ and $\mathcal{A}$, respectively; and let $p_i(s'|s,a)$ denote transition probabilities for example $i$. A discount factor on rewards (if any) is absorbed into the transition probabilities. Our cost (negative reward) functions are learned from supervised trajectories to produce policies that mimic the demonstrated behavior. Policies are described by $\mu \in \mathcal{G}$, where $\mathcal{G}$ is the space of all state-action frequency counts. In the case of deterministic planning, $\mu$ is simply an indicator variable denoting whether the state-action $s, a$ transition is encountered in the optimal policy. In the following, we use $\mathcal{M}$ both to denote a particular MDP, as well as to refer to the set of all state-action pairs in that MDP.

We hypothesize the existence of a base feature space $\mathcal{X}$ from which all other features are derived. A cost function over an MDP $\mathcal{M}$ is defined through this space as $c(f_{\mathcal{M}})$, where $f_{\mathcal{M}} : \mathcal{M} \rightarrow \mathcal{X}$ denotes a mapping from state-action pairs to points in base feature space, and $c$ is a cost function over $\mathcal{X}$. Intuitively, each state-action pair in the MDP has an associated feature vector, and the cost of that state-action pair is a function of that vector.

The input to the linear MMP algorithm is a set of training instances $\mathcal{D} = \{(\mathcal{M}_i, p_i, F_i, \mu_i, l_i)\}_{i=1}^n$. Each training instance consists of an MDP with transition probabilities $p_i$ and state-action pairs $(s_i, a_i) \in \mathcal{M}_i$ over which $d$-dimensional vectors of features mapped from the base feature space $\mathcal{X}$ are placed in the form of a $d \times |\mathcal{M}|$ feature matrix $F_i$. In linear MMP, $F_i$ is related to $c$ above by $[w^T F]^{s,a} = c(f_{\mathcal{M}_i}(s,a))$.

$\mu_i$ denotes the desired trajectory (or full policy) that exemplifies behavior we hope to match. The loss vector $l_i$ is a vector on the state-action pairs that indicates the loss for failing to match the demonstrated trajectory $\mu_i$. Typically, in this work we use a simple loss function that is 0 on all states occupied in the example trajectory and 1 elsewhere.

We use subscripts to denote indexing by training instance, and reserve superscripts for indexing into vectors. (E.g. $\mu_i^{s,a}$ is the expected state-action frequency for state $s$ and action $a$ of example $i$.) It is useful for some problems, such as robot path planning, to imagine representing the features as a set of maps and example paths through those maps. For instance, one feature map might indicate the elevation at each state, another the slope, and a third the presence of vegetation.

## 3   Theory

We discuss briefly the linear MMP regularized risk function as derived in (Ratliff et al., 2006) and provide the subgradient formula. We then present an intuitive and algorithmic exposition on the boosted version of this algorithm we use to learn a nonlinear cost function. The precise derivation of this algorithm is available as an appendix to the extended version of the paper, which can be found on the author's website.

### 3.1   The Maximum Margin Planning risk function

Crucial to the Maximum Margin Planning (MMP) approach is the development of a convex, but non-differentiable regularized risk function for the general margin or slack scaled (Tsochantaridis et al., 2005) maximum margin structured prediction problem. In (Ratliff et al., 2006), the authors

show that a subgradient descent procedure on this objective function can utilize efficient inference techniques resulting in an algorithm that is tractable in both computation and memory for large problems.

The risk function under this framework is

$$R(w) = \frac{1}{n} \sum_{i=1}^{n} \beta_i \left( w^T F_i \mu_i - \min_{\mu \in \mathcal{G}_i} (w^T F_i - l_i^T) \mu \right) + \frac{\lambda}{2} \|w\|^2,$$

which gives the following subgradient with respect to $w$

$$g_w = \frac{1}{n} \sum_{i=1}^{n} F_i \Delta^w \mu_i + \lambda w,$$

Here $F_i$ is the current set of learned features over example $i$, $\mu^* = \arg\min_{\mu \in \mathcal{G}_i} (w^T F_i - l_i^T) \mu$ and $\Delta^w \mu_i = \mu^* - \mu_i$. This latter expression points out that, intuitively, the subgradient compares the state-action visitation frequency counts between the example policy and the optimal policy with respect to the current reward function $w^T F_i$. The algorithm in its most basic form is given by the update rule $w_{t+1} \leftarrow w_t - \gamma_t g_t$, where $\{\gamma_t\}_{t=1}^{\infty}$ is a prespecified stepsize sequence and $g_t$ is a subgradient at the current timestep $t$.

Note that computing the subgradient requires solving the problem $\mu^* = \arg\min_{\mu \in \mathcal{G}_i} (w^T F_i - l_i^T) \mu$ for each MDP. This is precisely the problem of solving the particular MDP with the cost function $w^T F_i - l_i^T$, and can be implemented efficiently via a myriad of specialized algorithms such as $A^*$ in the context of planning.

## 3.2 Structured boosting of MMP

Maximum margin planning in its original formulation assumed the cost map is a linear function of a set of prespecified features. This is arguably the most restrictive assumption made in this framework. Similar to many machine learning algorithms, we find in practice substantial effort is put into choosing these features well. In this section, we describe at an intuitive and algorithmic level a boosting procedure for learning a nonlinear function of our base features. For clarity of exposition, a full derivation in terms of the functional gradient descent view of boosting (Mason et al., 1999) is postponed to the appendix of the extended version of this paper (available from the author's website). We encourage the reader to review this derivation as it differs in flavor from those previously seen in the literature in ways important to its application to general structured prediction problems.

This gradient boosting framework serves as a reduction (Beygelzimer et al., 2005) from the problem of finding good features for structured prediction to a problem of simple classification. At a high level, this algorithm learns a new feature by learning a classifier that is best correlated with the changes we would like to have made to locally decrease the loss had we an infinite number of parameters at our disposal.

In the case of MMPBOOST , this forms the following algorithm which is iterated:

- Fit the current model (using the current features) and compute the resulting loss-augmented cost map.

- Run the planner over this loss-augmented cost map to get the best loss-augmented path. Presumably, when the current feature set is not yet expressive enough, this path will differ significantly from the example path.

- Form positive examples by gathering feature vectors encountered along this loss-augmented path $\{(x_{\text{planned}}^{(i)}, 1)\}$ and form negative examples by gathering feature vectors encountered along the example path $\{(x_{\text{example}}^{(j)}, -1)\}$.

- Learn a classifier using this data set to generalize these suggestions to other points on the map.

- Apply this classifier to every cell of all example maps and add the result as a new feature to the feature matrix.

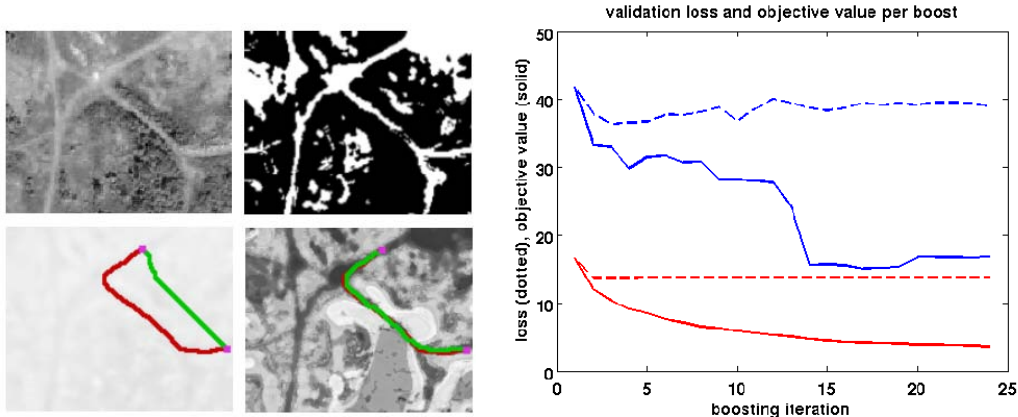

Figure 1: The four subimages to the left show (clockwise from upper left) a grayscale image used as base features for a hold out region, the first boosted feature learned by boosted MMP for this region, the results of boosted MMP on an example over this region (example red, learned path green), and the best linear fit of this limited feature set. The plot on the right compares boosting objective function value (red) and loss on a hold out set (blue) per boosting iteration between linear MMP (dashed) and boosted MMP (solid).

This simple procedure forms the MMPBOOST algorithm. If the original set of features cannot correctly represent as a linear function the cost variation necessary to explain the decisions made by the trainer, this algorithm tries to find a new feature as a nonlinear function of the original base set of features, that can best simultaneously raise the cost of the current erroneous path, and lower the cost of the example path. Importantly, this function takes the form of a classifier that can generalize this information to each cell of every map. Adding this feature to the current feature set provides an incremental step toward explaining the decisions made in the example paths.

# 4  Applications

In this section we demonstrate on three diverse problems how MMPBOOST improves performance in navigation and planning tasks.

## 4.1  Imitation Learning for Path Planning

We first consider a problem of learning to imitate example paths drawn by humans on publicly available overhead imagery. In this experiment, a teacher demonstrates optimal paths between a set of start and goal points on the image, and we compare the performance of MMPBOOST to that of a linear MMP algorithm in learning to imitate the behavior. The base features for this experiment consisted of the raw grayscale image, 5 Gaussian convolutions of it with standard deviations 1, 3, 5, 7, and 9, and a constant feature. Cost maps were created as a linear combination of these features in the case of MMP, and as a nonlinear function of these features in the case of MMPBOOST . The planner being trained was an 8-connected implementation of A*.

The results of these experiments are shown in Figure 1. The upper right panel on the left side of that Figure shows the grayscale overhead image of the holdout region used for testing. The training region was similar in nature, but taken over a different location. The features are particularly difficult for MMP since the space of cost maps it considers for this problem consists of only linear combinations of the same image at different resolutions. e.g. imagine taking various blurred versions of an image and trying to combine them to make any reasonable cost map. The lower left panel on the left side of Figure 1 shows that the best cost map MMP was able to find within this space was largely just a map with uniformly high cost everywhere. The learned cost map was largely uninformative causing the planner to choose the straight-line path between endpoints.

The lower right panel on the left side of Figure 1 shows the result of MMPBOOST on this problem on a holdout image of an area similar to that on which we trained. In this instance, we used regression trees with 10 terminal nodes as our dictionary $\mathcal{H}$, and trained them on the base features to match the functional gradient as described in Section 3.2. Since MMPBOOST searches through a space

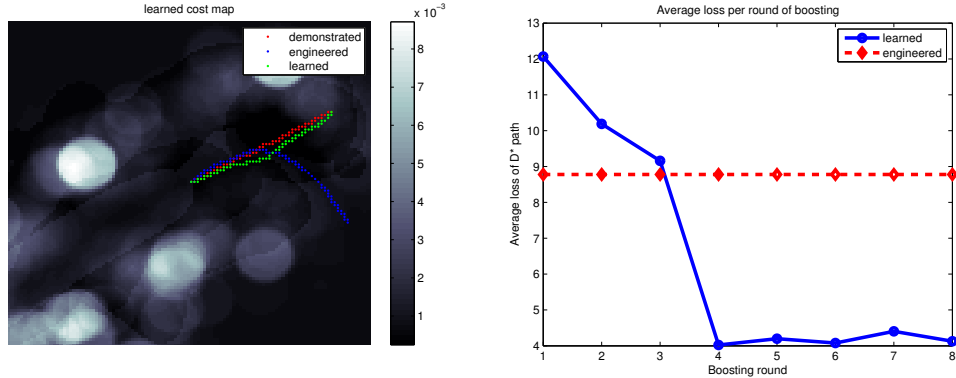

Figure 2: Left: An example learned cost map for a narrow path through trees showing the engineered system (blue line) wanting to take a short cut to the goal by veering off into dense woods instead of staying on the path as the human demonstrated (green line). In the learned cost map several boosted features combine to make the lowest-cost path (red line) match the human's preference of staying on the path. The robot is currently located in the center of the cost map. Right: A graph of the average A* path loss over the examples in each round of boosting. In just a few rounds the learned system exceeds the performance of the carefully engineered system.

nonlinear cost functions, it is able to perform significantly better than the linear MMP. Interestingly, the first feature it learned to explain the supervised behavior was to a large extent a road detection classifier. The right panel of Figure 1 compares plots of the objective value (red) and the loss on the holdout set (blue) per iteration between the linear MMP (dashed) and MMPBOOST (solid).

The first feature shown in figure 1 is interesting in that it largely represents the result of a path detector. The boosting algorithm chooses positive examples along the example path, and negative examples along the loss-augmented path, which are largely disjoint from the example paths. Surprisingly, MMPBOOST also outperformed linear MMP applied to additional features that were hand-engineered for this imagery. In principle, given example plans, MMPBOOST can act as a sophisticated image processing technique to transform any overhead (e.g. satellite) image directly to a cost map with no human intervention and feature engineering.

## 4.2 Learning from Human Driving Demonstration

We next consider the problem of learning to mimic human driving of an autonomous robot in complex outdoor and off-road terrain. We assume that a coarse *global planning* problem has been solved using overhead imagery and the MMPBOOST application presented above. Instead, we use MMP-BOOST to learn an local obstacle detection/avoidance system.

We consider the local region around the vehicle's position at time $t$ separately from the larger global environment. Our goal is to use the vehicle's onboard sensors to detect obstacles which were not visible in the overhead imagery or were not present when the imagery was collected. The onboard sensor suite used consists of ladar scanners to provide structural information about the environment, and color and NIR cameras to provide appearance information. From these sensors we compute a set of base features for each cell in a discretized 2-D map of the local area. These base features include quantities such as the estimated elevation and slope of the ground plane in the cell, and the average color and density of ladar points in the cell for various height ranges above the estimated ground plane. As training data we use logged sensor data from several kilometers of teleoperation of a large mobile robot through a challenging outdoor environment by an experienced operator.

In the previous example, the algorithm had access to both the complete path demonstrated by the teacher, and the same input data (overhead image) the teacher used while generating the path. However, in this example not only is the input data different (since the teacher generally controls the robot from behind and to the side using their own prior knowledge of the environment and highly capable vision system), but we face the additional challenge of estimating the path planned by the teacher at a particular time step from the vehicle motion we observe in future time steps, when the teacher is using additional data.

For this experiment we assume that the next 10 m of the path driven by the vehicle after time $t$ matches the operator's intended path at time $t$, and only compute loss over that section of the path. In practice this means that we create a set of local examples from each teleoperated path by sampling the internal state of the robot at discrete points in time. At each time $t$ we record the feature map generated by the robots onboard sensors of the local 10 m radius area surrounding it as well as the path the robot followed to the boundary of that area. Additionally, we model the operator's prior knowledge of the environment and their sensing of obstacles beyond the 10 m range by using our global planning solution to generate the minimum path costs from a set of points on the boundary of each local map to the global goal. The operator also attempted to match the range at which he reacted to obstacles not visible in the overhead data (such as vehicles that were placed in the robot's path) with the 10 m radius of the local map. An 8-connected variant of A* then chooses a path to one of the points on the boundary of the local map that minimizes the sum of costs accumulated along the path to the boundary point with the cost-to-goal from the boundary point to the goal. Using 8 terminal node classification trees as our dictionary $\mathcal{H}$, we then apply the MMPBOOST algorithm to determine transformations from base features to local costs so that the local trajectories executed by the human are chosen by the planner with large margin over all the other possible local trajectories.

The results of running MMPBOOST on the 301 examples in our data set are compared to the results given by the current human engineered cost production system used on the robot in Figure 2. The engineered system is the result of many man-hours of parameter tunning over weeks of field testing. The learned system started with the engineered feature maps, and then boosted in additional features as necessary. After just a few iterations of boosting the learned system displays significantly lower average loss than the engineered system, and corrects important navigational errors such as the one shown.

## 4.3    Learning a Fast Planner from a Slower one

Legged robots have unique capabilities not found in many mobile robots. In particular, they can step over or onto obstacles in their environment, allowing them to traverse complicated terrain. Algorithms have been developed which plan for foot placement in these environments, and have been successfully used on several biped robots (Chestnutt et al., 2005). In these cases, the planner evaluates various steps the robot can execute, to find a sequence of steps that is safe and is within the robot's capabilities. Another approach to legged robot navigation uses local techniques to reactively adjust foot placement while following a predefined path (Yagi & Lumelsky, 1999). This approach can fall into local minima or become stuck if the predefined path does not have valid footholds along its entire length.

Footstep planners have been shown to produce very good footstep sequences allowing legged robots to efficiently traverse a wide variety of terrain. This approach uses much of the robot's unique abilities, but is more computationally expensive than traditional mobile robot planners. Footstep planning occurs in a high-dimensional state space and therefore is often too computationally burdensome to be used for real-time replanning, limiting its scope of application to largely static environments. For most applications, the footstep planner implicitly solves a low dimensional navigational problem simultaneously with the footstep placement problem. Using MMPBOOST , we use body trajectories produced by the footstep planner to learn the nuances of this navigational problem in the form of a 2.5-dimensional navigational planner that can reproduce these trajectories. We are training a simple, navigational planner to effectively reproduce the body trajectories that typically result from a sophisticated footstep planner. We could use the resulting navigation planner in combination with a reactive solution (as in (Yagi & Lumelsky, 1999)). Instead, we pursue a hybrid approach of using the resulting simple planner as a heuristic to guide the footstep planner.

Using a 2-dimensional robot planner as a heuristic has been shown previously (Chestnutt et al., 2005) to dramatically improve planning performance, but the planner must be manually tuned to provide costs that serve as reasonable approximations of the true cost. To combat these computational problems we focus on the heuristic, which largely defines the behavior of the A* planner. Poorly informed admissible heuristics can cause the planner to erroneously attempt numerous dead ends before happening upon the optimal solution. On the other hand, well informed inadmissible heuristics can pull the planner quickly toward a solution whose cost, though suboptimal, is very close to the minimum. This lower-dimensional planner is then used in the heuristic to efficiently and

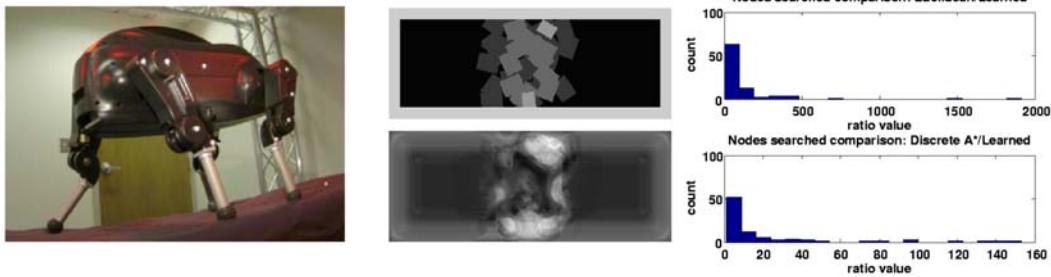

Figure 3: Left is an image of the robot used for the quadruped experiments. The center pair of images shows a typical height map (top), and the corresponding learned cost map (bottom) from a holdout set of the biped planning experiments. Notice how platform-like regions are given low costs toward the center but higher costs toward the edges, and the learned features interact to lower cost chutes to direct the planner through complicated regions. Right are two histograms showing the ratio distribution of the speed of both the admissible Euclidean (top) and the engineered heuristic (bottom) over an uninflated MMPBOOST heuristic on a holdout set of 90 examples from the biped experiment. In both cases, the MMPBOOST heuristic was uniformly better in terms of speed.

| | cost diff | | speedup | | cost diff | | speedup | |
|---|---|---|---|---|---|---|---|---|
| | mean | std | mean | std | mean | std | mean | std |
| | biped admissible | | | | biped inflated | | | |
| MMPBOOST vs Euclidean | 0.91 | 10.08 | 123.39 | 270.97 | 9.82 | 11.78 | 10.55 | 17.51 |
| MMPBOOST vs Engineered | -0.69 | 6.7 | 20.31 | 33.11 | 2.55 | 6.82 | 11.26 | 32.07 |
| | biped best-first | | | | quadruped inflated | | | |
| MMPBOOST vs Euclidean | -609.66 | 5315.03 | 272.99 | 1601.62 | 3.69 | 7.39 | 2.19 | 2.24 |
| MMPBOOST vs Engineered | 3.42 | 37.97 | 6.4 | 17.85 | -4.34 | 8.93 | 3.51 | 4.11 |

Figure 4: Statistics comparing the MMPBOOST heuristic to both a Euclidean and discrete navigational heuristic. See the text for descriptions of the values.

intelligently guide the footstep planner toward the goal, effectively displacing a large portion of the computational burden.

We demonstrate our results in both simulations and real-world experiments. Our procedure is to run a footstep planner over a series of randomly drawn two-dimensional terrain height maps that describe the world the robot is to traverse. The footstep planner produces trajectories of the robot from start to goal over the terrain map. We then apply MMPBOOST again using regression trees with 10 terminal nodes as the base classifier to learn cost features and weights that turn height maps into cost functions so that a 2-dimensional planner over the cost map mimics the body trajectory. We apply the planner to two robots: first the HRP-2 biped robot and second the LittleDog[2] quadruped robot.The quadruped tests were demonstrated on the robot.[3]

Figure 4 shows the resulting computational speedups (and the performance gains) of planning with the learned MMPBOOST heuristic over two previously implemented heuristics: a simple Euclidean heuristic that estimates the cost-to-go as the straight-line distance from the current state to the goal; and an alternative 2-dimensional navigational planner whose cost map was hand engineered. We tested three different versions of the planning configuration: (1) no inflation, in which the heuristic is expected to give its best approximation of the exact cost so that the heuristics are close to admissible (Euclidean is the only one who is truly admissible); (2) inflated, in which the heuristics are inflated by approximately 2.5 (this is the setting commonly used in practice for these planners); and (3) Best-first search, in which search nodes are expanded solely based on their heuristic value. The cost diff column relates on average the extent to which the cost of planning under the MMPBOOST heuristic is above or below the opposing heuristic. Loosely speaking this indicates how many more footsteps are taken under the MMPBOOST heuristic, i.e. negative values support MMPBOOST . The speedup column relates the average ratio of total nodes searched between the heuristics. In this case, large values are better, indicating the factor by which MMPBOOST outperforms its competition.

The most direct measure of heuristic performance arguably comes from the best-first search results. In this case, both the biped and quadruped planner using the learned heuristic significantly outperform their counterparts under a Euclidean heuristic.[4] While Euclidean often gets stuck for long periods of time in local minima, both the learned heuristic and to a lesser extent the engineered heuristic are able to navigate efficiently around these pitfalls. We note that A* biped performance gains were considerably higher: we believe this is because orientation plays a large role in planning for the quadruped.

## 5 Conclusions and Future Work

MMPBOOST combines the powerful ideas of structured prediction and functional gradient descent enabling learning by demonstration for a wide variety of applications. Future work will include extending the learning of mobile robot path planning to more complex configuration spaces that allow for modeling of vehicle dynamics. Further, we will pursue applications of the gradient boosting approach to other problems of structured prediction.

### Acknowledgments

The authors gratefully acknowledge the partial support of this research by the DARPA Learning for Locomotion and UPI contracts, and thank John Langford for enlightening conversations on reduction of structured learning problems.

## Footnotes

[1] Alternatively, all of these methods admit straightforward kernelization allowing the implicit learning within a Reproducing Kernel Hilbert space, but these kernel versions can be extremely memory and computationally intensive.

[2]Boston Dynamics designed the robot and provided the motion capture system used in the tests.

[3]A video demonstrating the robot walking across a terrain board is provided with this paper.

[4]The best-first quadruped planner under the MMPBOOST heuristic is on average approximately 1100 times faster than under the Euclidean heuristic in terms of the number of nodes searched.

## References

Beygelzimer, A., Dani, V., Hayes, T., Langford, J., & Zadrozny, B. (2005). Error limiting reductions between classification tasks. *ICML '05*. New York, NY.

Chestnutt, J., Lau, M., Cheng, G., Kuffner, J., Hodgins, J., & Kanade, T. (2005). Footstep planning for the Honda ASIMO humanoid. *Proceedings of the IEEE International Conference on Robotics and Automation*.

Dietterich, T. G., Ashenfelter, A., & Bulatov, Y. (2004). Training conditional random fields via gradient tree boosting. *ICML '04*.

Friedman, J. H. (1999a). Greedy function approximation: A gradient boosting machine. *Annals of Statistics*.

Hassani, S. (1998). *Mathematical physics*. Springer.

Mason, L., J.Baxter, Bartlett, P., & Frean, M. (1999). Functional gradient techniques for combining hypotheses. *Advances in Large Margin Classifiers*. MIT Press.

Ratliff, N., Bagnell, J. A., & Zinkevich, M. (2006). Maximum margin planning. *Twenty Second International Conference on Machine Learning (ICML06)*.

Taskar, B., Chatalbashev, V., Guestrin, C., & Koller, D. (2005). Learning structured prediction models: A large margin approach. *Twenty Second International Conference on Machine Learning (ICML05)*.

Taskar, B., Guestrin, C., & Koller, D. (2003). Max margin markov networks. *Advances in Neural Information Processing Systems (NIPS-14)*.

Tsochantaridis, I., Joachims, T., Hofmann, T., & Altun, Y. (2005). Large margin methods for structured and interdependent output variables. *Journal of Machine Learning Research*, 1453–1484.

Yagi, M., & Lumelsky, V. (1999). Biped robot locomotion in scenes with unknown obstacles. *Proceedings of the IEEE International Conference on Robotics and Automation* (pp. 375–380). Detroit, MI.

